# Inference in Multilayer Networks via Large Deviation Bounds

**Michael Kearns and Lawrence Saul**
AT&T Labs — Research
Shannon Laboratory
180 Park Avenue A-235
Florham Park, NJ 07932
{mkearns,lsaul}@research.att.com

## Abstract

We study probabilistic inference in large, layered Bayesian networks represented as directed acyclic graphs. We show that the intractability of exact inference in such networks does not preclude their effective use. We give algorithms for approximate probabilistic inference that exploit averaging phenomena occurring at nodes with large numbers of parents. We show that these algorithms compute rigorous lower and upper bounds on marginal probabilities of interest, prove that these bounds become exact in the limit of large networks, and provide rates of convergence.

## 1  Introduction

The promise of neural computation lies in exploiting the information processing abilities of simple computing elements organized into large networks. Arguably one of the most important types of information processing is the capacity for probabilistic reasoning.

The properties of *undirected* probabilistic models represented as symmetric networks have been studied extensively using methods from statistical mechanics (Hertz et al, 1991). Detailed analyses of these models are possible by exploiting averaging phenomena that occur in the thermodynamic limit of large networks.

In this paper, we analyze the limit of large, multilayer networks for probabilistic models represented as *directed* acyclic graphs. These models are known as *Bayesian networks* (Pearl, 1988; Neal, 1992), and they have different probabilistic semantics than symmetric neural networks (such as Hopfield models or Boltzmann machines). We show that the intractability of exact inference in multilayer Bayesian networks

does not preclude their effective use. Our work builds on earlier studies of variational methods (Jordan et al, 1997). We give algorithms for approximate probabilistic inference that exploit averaging phenomena occurring at nodes with $N \gg 1$ parents. We show that these algorithms compute rigorous lower and upper bounds on marginal probabilities of interest, prove that these bounds become exact in the limit $N \to \infty$, and provide rates of convergence.

## 2  Definitions and Preliminaries

A Bayesian network is a *directed* graphical probabilistic model, in which the nodes represent random variables, and the links represent causal dependencies. The joint distribution of this model is obtained by *composing* the local conditional probability distributions (or *tables*), **Pr**[child|parents], specified at each node in the network. For networks of binary random variables, so-called *transfer functions* provide a convenient way to parameterize conditional probability tables (CPTs). A transfer function is a mapping $f : [-\infty, \infty] \to [0, 1]$ that is everywhere differentiable and satisfies $f'(x) \geq 0$ for all $x$ (thus, $f$ is nondecreasing). If $f'(x) \leq \alpha$ for all $x$, we say that $f$ has *slope* $\alpha$. Common examples of transfer functions of bounded slope include the sigmoid $f(x) = 1/(1 + e^{-x})$, the cumulative gaussian $f(x) = \int_{-\infty}^{x} dt\, e^{-t^2}/\sqrt{\pi}$, and the noisy-OR $f(x) = 1 - e^{-x}$. Because the value of a transfer function $f$ is bounded between 0 and 1, it can be interpreted as the conditional probability that a binary random variable takes on a particular value. One use of transfer functions is to endow multilayer networks of soft-thresholding computing elements with probabilistic semantics. This motivates the following definition:

**Definition 1** *For a transfer function $f$, a **layered probabilistic $f$-network** has:*

- *Nodes representing binary variables $\{X_i^\ell\}$, $\ell = 1, \ldots, L$ and $i = 1, \ldots, N$. Thus, $L$ is the number of layers, and each layer contains $N$ nodes.*

- *For every pair of nodes $X_j^{\ell-1}$ and $X_i^\ell$ in adjacent layers, a real-valued weight $\theta_{ij}^{\ell-1}$ from $X_j^{\ell-1}$ to $X_i^\ell$.*

- *For every node $X_i^1$ in the first layer, a **bias** $p_i$.*

*We will sometimes refer to nodes in layer 1 as **inputs**, and to nodes in layer $L$ as **outputs**. A layered probabilistic $f$-network defines a joint probability distribution over all of the variables $\{X_i^\ell\}$ as follows: each input node $X_i^1$ is independently set to 1 with probability $p_i$, and to 0 with probability $1 - p_i$. Inductively, given binary values $X_j^{\ell-1} = x_j^{\ell-1} \in \{0, 1\}$ for all of the nodes in layer $\ell - 1$, the node $X_i^\ell$ is set to 1 with probability $f(\sum_{j=1}^{N} \theta_{ij}^{\ell-1} x_j^{\ell-1})$.*

Among other uses, multilayer networks of this form have been studied as hierarchical generative models of sensory data (Hinton et al, 1995). In such applications, the fundamental computational problem (known as *inference*) is that of estimating the marginal probability of evidence at some number of *output* nodes, say the first $K \leq N$. (The computation of conditional probabilities, such as diagnostic queries, can be reduced to marginals via Bayes rule.) More precisely, one wishes to estimate $\mathbf{Pr}[X_1^L = x_1, \ldots, X_K^L = x_K]$ (where $x_i \in \{0, 1\}$), a quantity whose exact computation involves an exponential sum over all the possible settings of the uninstantiated nodes in layers 1 through $L - 1$, and is known to be computationally intractable (Cooper, 1990).

## 3   Large Deviation and Union Bounds

One of our main weapons will be the theory of *large deviations*. As a first illustration of this theory, consider the input nodes $\{X_j^1\}$ (which are independently set to 0 or 1 according to their biases $p_j$) and the weighted sum $\sum_{j=1}^N \theta_{ij}^1 X_j^1$ that feeds into the $i$th node $X_i^2$ in the second layer. A typical large deviation bound (Kearns & Saul, 1997) states that for all $\epsilon > 0$, $\mathbf{Pr}[|\sum_{j=1}^N \theta_{ij}^1(X_j^1 - p_j)| > \epsilon] \le 2e^{-2\epsilon^2/(N\Theta^2)}$ where $\Theta$ is the largest weight in the network. If we make the scaling assumption that each weight $\theta_{ij}^1$ is bounded by $\tau/N$ for some constant $\tau$ (thus, $\Theta \le \tau/N$), then we see that the probability of large (order 1) deviations of this weighted sum from its mean decays exponentially with $N$. (Our methods can also provide results under the weaker assumption that all weights are bounded by $O(N^{-a})$ for $a > 1/2$.)

How can we apply this observation to the problem of inference? Suppose we are interested in the marginal probability $\mathbf{Pr}[X_i^2 = 1]$. Then the large deviation bound tells us that with probability at least $1 - \delta$ (where we define $\delta = 2e^{-2N\epsilon^2/\tau^2}$), the weighted sum at node $X_i^2$ will be within $\epsilon$ of its mean value $\mu_i = \sum_{j=1}^N \theta_{ij}^1 p_j$. Thus, with probability at least $1 - \delta$, we are assured that $\mathbf{Pr}[X_i^2 = 1]$ is at least $f(\mu_i - \epsilon)$ and at most $f(\mu_i + \epsilon)$. Of course, the flip side of the large deviation bound is that with probability at most $\delta$, the weighted sum may fall more than $\epsilon$ away from $\mu_i$. In this case we can make no guarantees on $\mathbf{Pr}[X_i^2 = 1]$ aside from the trivial lower and upper bounds of 0 and 1. Combining both eventualities, however, we obtain the overall bounds:

$$(1 - \delta)f(\mu_i - \epsilon) \; \le \; \mathbf{Pr}[X_i^2 = 1] \; \le \; (1 - \delta)f(\mu_i + \epsilon) + \delta. \tag{1}$$

Equation (1) is based on a simple *two-point* approximation to the distribution over the weighted sum of inputs, $\sum_{j=1}^N \theta_{ij}^1 X_j^1$. This approximation places one point, with weight $1 - \delta$, at either $\epsilon$ above or below the mean $\mu_i$ (depending on whether we are deriving the upper or lower bound); and the other point, with weight $\delta$, at either $-\infty$ or $+\infty$. The value of $\delta$ depends on the choice of $\epsilon$: in particular, as $\epsilon$ becomes smaller, we give more weight to the $\pm\infty$ point, with the trade-off governed by the large deviation bound. We regard the weight given to the $\pm\infty$ point as a *throw-away* probability, since with this weight we resort to the trivial bounds of 0 or 1 on the marginal probability $\mathbf{Pr}[X_i^2 = 1]$.

Note that the very simple bounds in Equation (1) already exhibit an interesting trade-off, governed by the choice of the parameter $\epsilon$—namely, as $\epsilon$ becomes smaller, the throw-away probability $\delta$ becomes larger, while the terms $f(\mu_i \pm \epsilon)$ converge to the same value. Since the overall bounds involve products of $f(\mu_i \pm \epsilon)$ and $1 - \delta$, the optimal value of $\epsilon$ is the one that balances this competition between probable explanations of the evidence and improbable deviations from the mean. This trade-off is reminiscent of that encountered between energy and entropy in mean-field approximations for symmetric networks (Hertz et al, 1991).

So far we have considered the marginal probability involving a single node in the second layer. We can also compute bounds on the marginal probabilities involving $K > 1$ nodes in this layer (which without loss of generality we take to be the nodes $X_1^2$ through $X_K^2$). This is done by considering the probability that *one or more* of the weighted sums entering these $K$ nodes in the second layer deviate by more than $\epsilon$ from their means. We can upper bound this probability by $K\delta$ by appealing to the so-called *union bound*, which simply states that the probability of a union of events is bounded by the sum of their individual probabilities. The union bound allows us to bound marginal probabilities involving multiple variables. For example,

consider the marginal probability $\mathbf{Pr}[X_1^2 = 1, \ldots, X_K^2 = 1]$. Combining the large deviation and union bounds, we find:

$$(1-K\delta)\prod_{i=1}^{K} f(\mu_i-\epsilon) \leq \mathbf{Pr}[X_1^2 = 1, \ldots, X_K^2 = 1] \leq (1-K\delta)\prod_{i=1}^{K} f(\mu_i+\epsilon)+K\delta. \quad (2)$$

A number of observations are in order here. First, Equation (2) directly leads to *efficient algorithms* for computing the upper and lower bounds. Second, although for simplicity we have considered $\epsilon$–deviations of the same size at each node in the second layer, the same methods apply to different choices of $\epsilon_i$ (and therefore $\delta_i$) at each node. Indeed, variations in $\epsilon_i$ can lead to significantly tighter bounds, and thus we exploit the freedom to choose different $\epsilon_i$ in the rest of the paper. This results, for example, in bounds of the form:

$$\left(1-\sum_{i=1}^{K}\delta_i\right)\prod_{i=1}^{K} f(\mu_i - \epsilon_i) \leq \mathbf{Pr}[X_1^2 = 1, \ldots, X_K^2 = 1], \quad \text{where} \quad \delta_i = 2e^{-2N\epsilon_i^2/\tau^2}. \quad (3)$$

The reader is invited to study the small but important differences between this lower bound and the one in Equation (2). Third, the arguments leading to bounds on the marginal probability $\mathbf{Pr}[X_1^2 = 1, \ldots, X_K^2 = 1]$ generalize in a straightforward manner to other patterns of evidence besides all 1's. For instance, again just considering the lower bound, we have:

$$\left(1-\sum_{i=1}^{K}\delta_i\right)\prod_{x_i=0}[1-f(\mu_i+\epsilon_i)]\prod_{x_i=1} f(\mu_i-\epsilon_i) \leq \mathbf{Pr}[X_1^2 = x_1, \ldots, X_K^2 = x_K] \quad (4)$$

where $x_i \in \{0,1\}$ are arbitrary binary values. Thus together the large deviation and union bounds provide the means to compute upper and lower bounds on the marginal probabilities over nodes in the second layer. Further details and consequences of these bounds for the special case of *two-layer* networks are given in a companion paper (Kearns & Saul, 1997); our interest here, however, is in the more challenging generalization to multilayer networks.

## 4 Multilayer Networks: Inference via Induction

In extending the ideas of the previous section to multilayer networks, we face the problem that the nodes in the second layer, unlike those in the first, are *not* independent. But we can still adopt an inductive strategy to derive bounds on marginal probabilities. The crucial observation is that *conditioned* on the values of the incoming weighted sums at the nodes in the second layer, the variables $\{X_i^2\}$ do become independent. More generally, conditioned on these weighted sums all falling "near" their means — an event whose probability we quantified in the last section — the nodes $\{X_i^2\}$ become "almost" independent. It is exactly this near-independence that we now formalize and exploit inductively to compute bounds for multilayer networks. The first tool we require is an appropriate generalization of the large deviation bound, which does not rely on precise knowledge of the means of the random variables being summed.

**Theorem 1** *For all $1 \leq j \leq N$, let $X_j \in \{0,1\}$ denote independent binary random variables, and let $|\tau_j| \leq \tau$. Suppose that the means are bounded by $|\mathbf{E}[X_j]-p_j| \leq \Delta_j$, where $0 < \Delta_j \leq p_j \leq 1 - \Delta_j$. Then for all $\epsilon > \frac{1}{N}\sum_{j=1}^{N} |\tau_j|\Delta_j$:*

$$\mathbf{Pr}\left[\left|\frac{1}{N}\sum_{j=1}^{N}\tau_j(X_j - p_j)\right| > \epsilon\right] \leq 2e^{-\frac{2N}{\tau^2}\left(\epsilon-\frac{1}{N}\sum_{j=1}^{N} |\tau_j|\Delta_j\right)^2}. \quad (5)$$

The proof of this result is omitted due to space considerations. Now for induction, consider the nodes in the $\ell$th layer of the network. Suppose we are told that for *every* $i$, the weighted sum $\sum_{j=1}^{N} \theta_{ij}^{\ell-1} X_j^{\ell-1}$ entering into the node $X_i^{\ell}$ lies in the interval $[\mu_i^{\ell} - \epsilon_i^{\ell}, \mu_i^{\ell} + \epsilon_i^{\ell}]$, for some choice of the $\mu_i^{\ell}$ and the $\epsilon_i^{\ell}$. Then the mean of node $X_i^{\ell}$ is constrained to lie in the interval $[p_i^{\ell} - \Delta_i^{\ell}, p_i^{\ell} + \Delta_i^{\ell}]$, where

$$p_i^{\ell} = \frac{1}{2} \left[ f(\mu_i^{\ell} - \epsilon_i^{\ell}) + f(\mu_i^{\ell} + \epsilon_i^{\ell}) \right] \tag{6}$$

$$\Delta_i^{\ell} = \frac{1}{2} \left[ f(\mu_i^{\ell} + \epsilon_i^{\ell}) - f(\mu_i^{\ell} - \epsilon_i^{\ell}) \right] . \tag{7}$$

Here we have simply run the leftmost and rightmost allowed values for the incoming weighted sums through the transfer function, and defined the interval around the mean of unit $X_i^{\ell}$ to be centered around $p_i^{\ell}$. Thus we have translated uncertainties on the incoming weighted sums to layer $\ell$ into conditional uncertainties on the means of the nodes $X_i^{\ell}$ in layer $\ell$. To complete the cycle, we now translate these into conditional uncertainties on the incoming weighted sums to layer $\ell + 1$. In particular, conditioned on the original intervals $[\mu_i^{\ell} - \epsilon_i^{\ell}, \mu_i^{\ell} + \epsilon_i^{\ell}]$, what is probability that for each $i$, $\sum_{j=1}^{N} \theta_{ij}^{\ell} X_j^{\ell}$ lies inside some new interval $[\mu_i^{\ell+1} - \epsilon_i^{\ell+1}, \mu_i^{\ell+1} + \epsilon_i^{\ell+1}]$? In order to make some guarantee on this probability, we set $\mu_i^{\ell+1} = \sum_{j=1}^{N} \theta_{ij}^{\ell} p_j^{\ell}$ and assume that $\epsilon_i^{\ell+1} > \sum_{j=1}^{N} |\theta_{ij}^{\ell}| \Delta_j^{\ell}$. These conditions suffice to ensure that the new intervals contain the (conditional) *expected values* of the weighted sums $\sum_{j=1}^{N} \theta_{ij}^{\ell} X_j^{\ell}$, and that the new intervals are large enough to encompass the incoming uncertainties. Because these conditions are a minimal requirement for establishing any probabilistic guarantees, we shall say that the $[\mu_i^{\ell} - \epsilon_i^{\ell}, \mu_i^{\ell} + \epsilon_i^{\ell}]$ define a *valid set of $\epsilon$-intervals* if they meet these conditions for all $1 \le i \le N$. Given a valid set of $\epsilon$-intervals at the $(\ell + 1)$th layer, it follows from Theorem 1 and the union bound that the weighted sums entering nodes in layer $\ell + 1$ obey

$$\mathbf{Pr} \left[ \left| \sum_{j=1}^{N} \theta_{ij}^{\ell} X_j^{\ell} - \mu_i^{\ell+1} \right| > \epsilon_i^{\ell+1} \text{ for some } 1 \le i \le N \right] \le \sum_{i=1}^{N} \delta_i^{\ell+1} \tag{8}$$

where

$$\delta_i^{\ell+1} = 2 e^{-\frac{2N}{r^2} \left( \epsilon_i^{\ell+1} - \sum_{j=1}^{N} |\theta_{ij}^{\ell}| \Delta_j^{\ell} \right)^2} . \tag{9}$$

In what follows, we shall frequently make use of the fact that the weighted sums $\sum_{j=1}^{N} \theta_{ij}^{\ell} X_i^{\ell}$ are bounded by intervals $[\mu_i^{\ell+1} - \epsilon_i^{\ell+1}, \mu_i^{\ell+1} + \epsilon_i^{\ell+1}]$. This motivates the following definitions.

**Definition 2** *Given a valid set of $\epsilon$-intervals and binary values $\{X_i^{\ell} = x_i^{\ell}\}$ for the nodes in the $\ell$th layer, we say that the $(\ell + 1)$st layer of the network* **satisfies** *its $\epsilon$-intervals if $\left| \sum_{j=1}^{N} \theta_{ij}^{\ell} x_j^{\ell} - \mu_i^{\ell+1} \right| < \epsilon^{\ell+1}$ for all $1 \le i \le N$. Otherwise, we say that the $(\ell + 1)$st layer* **violates** *its $\epsilon$-intervals.*

Suppose that we are given a valid set of $\epsilon$-intervals and that we sample from the joint distribution defined by the probabilistic $f$-network. The right hand side of Equation (8) provides an upper bound on the conditional probability that the $(\ell + 1)$st layer violates its $\epsilon$-intervals, given that the $\ell$th layer did not. This upper bound may be vacuous (that is, larger than 1), so let us denote by $\delta^{\ell+1}$ whichever is smaller — the right hand side of Equation (8), or 1; in other words, $\delta^{\ell+1} = \min\left\{ \sum_{i=1}^{N} \delta_i^{\ell+1}, 1 \right\}$. Since at the $\ell$th layer, the probability of violating the $\epsilon$-intervals is at most $\delta^{\ell}$ we

are guaranteed that with probability at least $\prod_{\ell>1}[1-\delta^\ell]$, *all* the layers satisfy their $\epsilon$-intervals. Conversely, we are guaranteed that the probability that any layer violates its $\epsilon$-intervals is at most $1 - \prod_{\ell>1}[1-\delta^\ell]$. Treating this as a throw-away probability, we can now compute upper and lower bounds on marginal probabilities involving nodes at the $L$th layer exactly as in the case of nodes at the second layer. This yields the following theorem.

**Theorem 2** *For any subset* $\{X_1^L, \ldots, X_K^L\}$ *of the outputs of a probabilistic f-network, for any setting* $x_1, \ldots, x_K$, *and for any valid set of $\epsilon$-intervals, the marginal probability of partial evidence in the output layer obeys:*

$$\prod_{\ell>1} [1 - \delta^\ell] \prod_{x_i=1} f(\mu_i^L - \epsilon_i^L) \prod_{x_i=0} [1 - f(\mu_i^L + \epsilon_i^L)] \tag{10}$$

$$\leq \mathbf{Pr}[X_1^L = x_1, \ldots, X_K^L = x_K]$$

$$\leq \prod_{\ell>1} [1 - \delta^\ell] \prod_{x_i=1} f(\mu_i^L + \epsilon_i^L) \prod_{x_i=0} [1 - f(\mu_i^L - \epsilon_i^L)] + \left(1 - \prod_{\ell>1} [1 - \delta^\ell]\right) \tag{11}$$

Theorem 2 generalizes our earlier results for marginal probabilities over nodes in the second layer; for example, compare Equation (10) to Equation (4). Again, the upper and lower bounds can be efficiently computed for all common transfer functions.

## 5   Rates of Convergence

To demonstrate the power of Theorem 2, we consider how the gap (or additive difference) between these upper and lower bounds on $\mathbf{Pr}[X_1^L = x_1, \ldots, X_K^L = x_K]$ behaves for some crude (but informed) choices of the $\{\epsilon_i^\ell\}$. Our goal is to derive the *rate* at which these upper and lower bounds converge to the same value as we examine larger and larger networks. Suppose we choose the $\epsilon$-intervals inductively by defining $\Delta_i^1 = 0$ and setting

$$\epsilon_i^{\ell+1} = \sum_{j=1}^N |\theta_{ij}^\ell| \Delta_j^\ell + \sqrt{\frac{\gamma \tau^2 \ln N}{N}} \tag{12}$$

for some $\gamma > 1$. From Equations (8) and (9), this choice gives $\delta^{\ell+1} \leq 2N^{1-2\gamma}$ as an upper bound on the probability that the $(\ell+1)$th layer violates its $\epsilon$-intervals. Moreover, denoting the gap between the upper and lower bounds in Theorem 2 by $G$, it can be shown that:

$$G \leq 2\alpha\sqrt{\frac{\gamma \tau^2 \ln N}{N}} \left[\frac{1 - (\alpha\tau)^L}{1 - \alpha\tau}\right] \sum_{i=1}^K \prod_{\substack{v_j=1 \\ j \neq i}} f(\mu_j^L + \epsilon_j^L) \prod_{\substack{v_j=0 \\ j \neq i}} [1 - f(\mu_j^L - \epsilon_j^L)] + \frac{2L}{N^{2\gamma-1}}. \tag{13}$$

Let us briefly recall the definitions of the parameters on the right hand side of this equation: $\alpha$ is the maximal slope of the transfer function $f$, $N$ is the number of nodes in each layer, $K$ is the number of nodes with evidence, $\tau = N\Theta$ is $N$ times the largest weight in the network, $L$ is the number of layers, and $\gamma > 1$ is a parameter at our disposal. The first term of this bound essentially has a $1/\sqrt{N}$ dependence on $N$, but is multiplied by a damping factor that we might typically expect to decay exponentially with the number $K$ of outputs examined. To see this, simply notice that each of the factors $f(\mu_j + \epsilon_j)$ and $[1 - f(\mu_j - \epsilon_j)]$ is bounded by 1; furthermore,

since all the means $\mu_j$ are bounded, if $N$ is large compared to $\gamma$ then the $\epsilon_i$ are small, and each of these factors is in fact bounded by some value $\beta < 1$. Thus the first term in Equation (13) is bounded by a constant times $\beta^{K-1} K \sqrt{\ln(N)/N}$. Since it is natural to expect the marginal probability of interest itself to decrease exponentially with $K$, this is desirable and natural behavior.

Of course, in the case of large $K$, the behavior of the resulting overall bound can be dominated by the second term $2L/N^{2\gamma-1}$ of Equation (13). In such situations, however, we can consider larger values of $\gamma$, possibly even of order $K$; indeed, for sufficiently large $\gamma$, the first term (which scales like $\sqrt{\gamma}$) must necessarily overtake the second one. Thus there is a clear trade-off between the two terms, as well as optimal value of $\gamma$ that sets them to be (roughly) the same magnitude. Generally speaking, for fixed $K$ and large $N$, we observe that the difference between our upper and lower bounds on $\mathbf{Pr}[X_1^L = x_1, \ldots, X_K^L = x_K]$ vanishes as $O\left(\sqrt{\ln(N)/N}\right)$.

## 6   An Algorithm for Fixed Multilayer Networks

We conclude by noting that the specific choices made for the parameters $\epsilon_i$ in Section 5 to derive rates of convergence may be far from the optimal choices for a fixed network of interest. However, Theorem 2 directly suggests a natural algorithm for approximate probabilistic inference. In particular, regarding the upper and lower bounds on $\mathbf{Pr}[X_1^L = x_1, \ldots, X_K^L = x_K]$ as functions of $\{\epsilon_i^\ell\}$, we can optimize these bounds by standard numerical methods. For the upper bound, we may perform gradient descent in the $\{\epsilon_i^\ell\}$ to find a local minimum, while for the lower bound, we may perform gradient ascent to find a local maximum. The components of these gradients in both cases are easily computable for all the commonly studied transfer functions. Moreover, the constraint of maintaining valid $\epsilon$-intervals can be enforced by maintaining a floor on the $\epsilon$-intervals in one layer in terms of those at the previous one. The practical application of this algorithm to interesting Bayesian networks will be studied in future work.

## References

Cooper, G. (1990). Computational complexity of probabilistic inference using Bayesian belief networks. *Artificial Intelligence* **42**:393-405.

Hertz, J,. Krogh, A., & Palmer, R. (1991). *Introduction to the theory of neural computation*. Addison-Wesley, Redwood City, CA.

Hinton, G., Dayan, P., Frey, B., and Neal, R. (1995). The wake-sleep algorithm for unsupervised neural networks. *Science* **268**:1158–1161.

Jordan, M., Ghahramani, Z., Jaakkola, T., & Saul, L. (1997). An introduction to variational methods for graphical models. In M. Jordan, ed. *Learning in Graphical Models*. Kluwer Academic.

Kearns, M., & Saul, L. (1998). Large deviation methods for approximate probabilistic inference. In *Proceedings of the 14th Annual Conference on Uncertainty in Artificial Intelligence*.

Neal, R. (1992). Connectionist learning of belief networks. *Artificial Intelligence* **56**:71–113.

Pearl, J. (1988). *Probabilistic Reasoning in Intelligent Systems: Networks of Plausible Inference*. Morgan Kaufmann, San Mateo, CA.
